# Improving the Accuracy and Speed of Support Vector Machines

**Chris J.C. Burges**
Bell Laboratories
Lucent Technologies, Room 3G429
101 Crawford's Corner Road
Holmdel, NJ 07733-3030
burges@bell-labs.com

**Bernhard Schölkopf***
Max–Planck–Institut für
biologische Kybernetik,
Spemannstr. 38
72076 Tübingen, Germany
bs@mpik-tueb.mpg.de

## Abstract

Support Vector Learning Machines (SVM) are finding application
in pattern recognition, regression estimation, and operator inver-
sion for ill-posed problems. Against this very general backdrop,
any methods for improving the generalization performance, or for
improving the speed in test phase, of SVMs are of increasing in-
terest. In this paper we combine two such techniques on a pattern
recognition problem. The method for improving generalization per-
formance (the "virtual support vector" method) does so by incor-
porating known invariances of the problem. This method achieves
a drop in the error rate on 10,000 NIST test digit images of 1.4%
to 1.0%. The method for improving the speed (the "reduced set"
method) does so by approximating the support vector decision sur-
face. We apply this method to achieve a factor of fifty speedup in
test phase over the virtual support vector machine. The combined
approach yields a machine which is both 22 times faster than the
original machine, and which has better generalization performance,
achieving 1.1% error. The virtual support vector method is appli-
cable to any SVM problem with known invariances. The reduced
set method is applicable to any support vector machine.

## 1 INTRODUCTION

Support Vector Machines are known to give good results on pattern recognition
problems despite the fact that they do not incorporate problem domain knowledge.

However, they exhibit classification speeds which are substantially slower than those of neural networks (LeCun et al., 1995).

The present study is motivated by the above two observations. First, we shall improve accuracy by incorporating knowledge about invariances of the problem at hand. Second, we shall increase classification speed by reducing the complexity of the decision function representation. This paper thus brings together two threads explored by us during the last year (Schölkopf, Burges & Vapnik, 1996; Burges, 1996).

The method for incorporating invariances is applicable to any problem for which the data is expected to have known symmetries. The method for improving the speed is applicable to any support vector machine. Thus we expect these methods to be widely applicable to problems beyond pattern recognition (for example, to the regression estimation problem (Vapnik, Golowich & Smola, 1996)).

After a brief overview of Support Vector Machines in Section 2, we describe how problem domain knowledge was used to improve generalization performance in Section 3. Section 4 contains an overview of a general method for improving the classification speed of Support Vector Machines. Results are collected in Section 5. We conclude with a discussion.

## 2   SUPPORT VECTOR LEARNING MACHINES

This Section summarizes those properties of Support Vector Machines (SVM) which are relevant to the discussion below. For details on the basic SVM approach, the reader is referred to (Boser, Guyon & Vapnik, 1992; Cortes & Vapnik, 1995; Vapnik, 1995). We end by noting a physical analogy.

Let the training data be elements $\mathbf{x}_i \in \mathcal{L}, \mathcal{L} = \mathbf{R}^d, i = 1, \ldots, \ell$, with corresponding class labels $y_i \in \{\pm 1\}$. An SVM performs a mapping $\Phi : \mathcal{L} \to \mathcal{H}, \mathbf{x} \mapsto \bar{\mathbf{x}}$ into a high (possibly infinite) dimensional Hilbert space $\mathcal{H}$. In the following, vectors in $\mathcal{H}$ will be denoted with a bar. In $\mathcal{H}$, the SVM decision rule is simply a separating hyperplane: the algorithm constructs a decision surface with normal $\bar{\Psi} \in \mathcal{H}$ which separates the $\mathbf{x}_i$ into two classes:

$$\bar{\Psi} \cdot \bar{\mathbf{x}}_i + b \geq k_0 - \xi_i, \quad y_i = +1 \tag{1}$$
$$\bar{\Psi} \cdot \bar{\mathbf{x}}_i + b \leq k_1 + \xi_i, \quad y_i = -1 \tag{2}$$

where the $\xi_i$ are positive slack variables, introduced to handle the non-separable case (Cortes & Vapnik, 1995), and where $k_0$ and $k_1$ are typically defined to be $+1$ and $-1$, respectively. $\bar{\Psi}$ is computed by minimizing the objective function

$$\frac{\bar{\Psi} \cdot \bar{\Psi}}{2} + C(\sum_{i=1}^{\ell} \xi_i)^p \tag{3}$$

subject to (1), (2), where C is a constant, and we choose $p = 2$. In the separable case, the SVM algorithm constructs that separating hyperplane for which the margin between the positive and negative examples in $\mathcal{H}$ is maximized. A test vector $\mathbf{x} \in \mathcal{L}$ is then assigned a class label $\{+1, -1\}$ depending on whether $\bar{\Psi} \cdot \Phi(\mathbf{x}) + b$ is greater or less than $(k_0 + k_1)/2$. Support vectors $\mathbf{s}_j \in \mathcal{L}$ are defined as training samples for which one of Equations (1) or (2) is an equality. (We name the support vectors $\mathbf{s}$ to distinguish them from the rest of the training data). The solution $\bar{\Psi}$ may be expressed

$$\bar{\Psi} = \sum_{j=1}^{N_S} \alpha_j y_j \Phi(\mathbf{s}_j) \tag{4}$$

where $\alpha_j \geq 0$ are the positive weights, determined during training, $y_j \in \{\pm 1\}$ the class labels of the $\mathbf{s}_j$, and $N_S$ the number of support vectors. Thus in order to classify a test point $\mathbf{x}$ one must compute

$$\bar{\Psi} \cdot \bar{\mathbf{x}} = \sum_{j=1}^{N_S} \alpha_j y_j \bar{\mathbf{s}}_j \cdot \bar{\mathbf{x}} = \sum_{j=1}^{N_S} \alpha_j y_j \Phi(\mathbf{s}_j) \cdot \Phi(\mathbf{x}) = \sum_{j=1}^{N_S} \alpha_j y_j K(\mathbf{s}_j, \mathbf{x}). \qquad (5)$$

One of the key properties of support vector machines is the use of the kernel $K$ to compute dot products in $\mathcal{H}$ without having to explicitly compute the mapping $\Phi$.

It is interesting to note that the solution has a simple physical interpretation in the high dimensional space $\mathcal{H}$. If we assume that each support vector $\bar{\mathbf{s}}_j$ exerts a perpendicular force of size $\alpha_j$ and sign $y_j$ on a solid plane sheet lying along the hyperplane $\bar{\Psi} \cdot \bar{\mathbf{x}} + b = (k_0 + k_1)/2$, then the solution satisfies the requirements of mechanical stability. At the solution, the $\alpha_j$ can be shown to satisfy $\sum_{j=1}^{N_S} \alpha_j y_j = 0$, which translates into the forces on the sheet summing to zero; and Equation (4) implies that the torques also sum to zero.

## 3 IMPROVING ACCURACY

This section follows the reasoning of (Schölkopf, Burges, & Vapnik, 1996). Problem domain knowledge can be incorporated in two different ways: the knowledge can be directly built into the algorithm, or it can be used to generate artificial training examples ("virtual examples"). The latter significantly slows down training times, due to both correlations in the artificial data and to the increased training set size (Simard et al., 1992); however it has the advantage of being readily implemented for any learning machine and for any invariances. For instance, if instead of Lie groups of symmetry transformations one is dealing with discrete symmetries, such as the bilateral symmetries of Vetter, Poggio, & Bülthoff (1994), then derivative–based methods (e.g. Simard et al., 1992) are not applicable.

For support vector machines, an intermediate method which combines the advantages of both approaches is possible. The support vectors characterize the solution to the problem in the following sense: If all the other training data were removed, and the system retrained, then the solution would be unchanged. Furthermore, those support vectors $\bar{\mathbf{s}}_i$ which are not errors are close to the decision boundary in $\mathcal{H}$, in the sense that they either lie exactly on the margin ($\xi_i = 0$) or close to it ($\xi_i < 1$). Finally, different types of SVM, built using different kernels, tend to produce the same set of support vectors (Schölkopf, Burges, & Vapnik, 1995). This suggests the following algorithm: first, train an SVM to generate a set of support vectors $\{\mathbf{s}_1, \ldots, \mathbf{s}_{N_s}\}$; then, generate the artificial examples (*virtual support vectors*) by applying the desired invariance transformations to $\{\mathbf{s}_1, \ldots, \mathbf{s}_{N_s}\}$; finally, train another SVM on the new set. To build a ten–class classifier, this procedure is carried out separately for ten binary classifiers.

Apart from the increase in overall training time (by a factor of two, in our experiments), this technique has the disadvantage that many of the virtual support vectors become support vectors for the second machine, increasing the number of summands in Equation (5) and hence decreasing classification speed. However, the latter problem can be solved with the reduced set method, which we describe next.

## 4   IMPROVING CLASSIFICATION SPEED

The discussion in this Section follows that of (Burges, 1996). Consider a set of vectors $\mathbf{z}_k \in \mathcal{L}, k = 1, \ldots, N_Z$ and corresponding weights $\gamma_k \in \mathbf{R}$ for which

$$\bar{\Psi}' \equiv \sum_{k=1}^{N_Z} \gamma_k \Phi(\mathbf{z}_k) \tag{6}$$

minimizes (for fixed $N_Z$) the Euclidean distance to the original solution:

$$\rho = \|\bar{\Psi} - \bar{\Psi}'\|. \tag{7}$$

Note that $\rho$, expressed here in terms of vectors in $\mathcal{H}$, can be expressed entirely in terms of functions (using the kernel $K$) of vectors in the input space $\mathcal{L}$. The $\{(\gamma_k, \mathbf{z}_k) \mid k = 1, \ldots, N_Z\}$ is called the *reduced set*. To classify a test point $\mathbf{x}$, the expansion in Equation (5) is replaced by the approximation

$$\bar{\Psi}' \cdot \bar{\mathbf{x}} = \sum_{k=1}^{N_Z} \gamma_k \bar{\mathbf{z}}_k \cdot \bar{\mathbf{x}} = \sum_{k=1}^{N_Z} \gamma_k K(\mathbf{z}_k, \mathbf{x}). \tag{8}$$

The goal is then to choose the smallest $N_Z \ll N_S$, and corresponding reduced set, such that any resulting loss in generalization performance remains acceptable. Clearly, by allowing $N_Z = N_S$, $\rho$ can be made zero. Interestingly, there are non-trivial cases where $N_Z < N_S$ and $\rho = 0$, in which case the reduced set leads to an increase in classification speed with no loss in generalization performance. Note that reduced set vectors are not support vectors, in that they do not necessarily lie on the separating margin and, unlike support vectors, are not training samples.

While the reduced set can be found exactly in some cases, in general an unconstrained conjugate gradient method is used to find the $\mathbf{z}_k$ (while the corresponding optimal $\gamma_k$ can be found exactly, for all $k$). The method for finding the reduced set is computationally very expensive (the final phase constitutes a conjugate gradient descent in a space of $(d + 1) \cdot N_Z$ variables, which in our case is typically of order 50,000).

## 5   EXPERIMENTAL RESULTS

In this Section, by "accuracy" we mean generalization performance, and by "speed" we mean classification speed. In our experiments, we used the MNIST database of 60000+10000 handwritten digits, which was used in the comparison investigation of LeCun et al (1995). In that study, the error rate record of 0.7% is held by a boosted convolutional neural network ("LeNet4").

We start by summarizing the results of the virtual support vector method. We trained ten binary classifiers using $C = 10$ in Equation (3). We used a polynomial kernel $K(\mathbf{x}, \mathbf{y}) = (\mathbf{x} \cdot \mathbf{y})^5$. Combining classifiers then gave 1.4% error on the 10,000 test set; this system is referred to as ORIG below. We then generated new training data by translating the resulting support vectors by one pixel in each of four directions, and trained a new machine (using the same parameters). This machine, which is referred to as VSV below, achieved 1.0% error on the test set. The results for each digit are given in Table 1.

Note that the improvement in accuracy comes at a cost in speed of approximately a factor of 2. Furthermore, the speed of ORIG was comparatively slow to start with (LeCun et al., 1995), requiring approximately 14 million multiply adds for one

Table 1: Generalization Performance Improvement by Incorporating Invariances. $N_E$ and $N_{SV}$ are the number of errors and number of support vectors respectively; "ORIG" refers to the original support vector machine, "VSV" to the machine trained on virtual support vectors.

| Digit | $N_E$ ORIG | $N_E$ VSV | $N_{SV}$ ORIG | $N_{SV}$ VSV |
|-------|-----------|-----------|---------------|--------------|
| 0 | 17 | 15 | 1206 | 2938 |
| 1 | 15 | 13 | 757 | 1887 |
| 2 | 34 | 23 | 2183 | 5015 |
| 3 | 32 | 21 | 2506 | 4764 |
| 4 | 30 | 30 | 1784 | 3983 |
| 5 | 29 | 23 | 2255 | 5235 |
| 6 | 30 | 18 | 1347 | 3328 |
| 7 | 43 | 39 | 1712 | 3968 |
| 8 | 47 | 35 | 3053 | 6978 |
| 9 | 56 | 40 | 2720 | 6348 |

Table 2: Dependence of Performance of Reduced Set System on Threshold. The numbers in parentheses give the corresponding number of errors on the test set. Note that Thrsh Test gives a lower bound for these numbers.

| Digit | Thrsh VSV | Thrsh Bayes | Thrsh Test |
|-------|-----------|-------------|------------|
| 0 | 1.39606 (19) | 1.48648 (18) | 1.54696 (17) |
| 1 | 3.98722 (24) | 4.43154 (12) | 4.32039 (10) |
| 2 | 1.27175 (31) | 1.33081 (30) | 1.26466 (29) |
| 3 | 1.26518 (29) | 1.42589 (27) | 1.33822 (26) |
| 4 | 2.18764 (37) | 2.3727 (35) | 2.30899 (33) |
| 5 | 2.05222 (33) | 2.21349 (27) | 2.27403 (24) |
| 6 | 0.95086 (25) | 1.06629 (24) | 0.790952 (20) |
| 7 | 3.0969 (59) | 3.34772 (57) | 3.27419 (54) |
| 8 | -1.06981 (39) | -1.19615 (40) | -1.26365 (37) |
| 9 | 1.10586 (40) | 1.10074 (40) | 1.13754 (39) |

classification (this can be reduced by caching results of repeated support vectors (Burges, 1996)). In order to become competitive with systems with comparable accuracy, we will need approximately a factor of fifty improvement in speed. We therefore approximated VSV with a reduced set system RS with a factor of fifty fewer vectors than the number of support vectors in VSV.

Since the reduced set method computes an *approximation* to the decision surface in the high dimensional space, it is likely that the accuracy of RS could be improved by choosing a different threshold $b$ in Equations (1) and (2). We computed that threshold which gave the empirical Bayes error for the RS system, measured on the training set. This can be done easily by finding the maximum of the difference between the two un-normalized cumulative distributions of the values of the dot products $\bar{\Psi} \cdot \bar{x}_i$, where the $x_i$ are the *original* training data. Note that the effects of bias are reduced by the fact that VSV (and hence RS) was trained only on shifted data, and not on any of the original data. Thus, in the absence of a validation set, the original training data provides a reasonable means of estimating the Bayes threshold. This is a serendipitous bonus of the VSV approach. Table 2 compares results obtained using the threshold generated by the training procedure for the VSV system; the estimated Bayes threshold for the RS system; and, for comparison

Table 3: Speed Improvement Using the Reduced Set method. The second through fourth columns give numbers of errors on the test set for the original system, the virtual support vector system, and the reduced set system. The last three columns give, for each system, the number of vectors whose dot product must be computed in test phase.

| Digit | ORIG Err | VSV Err | RS Err | ORIG # SV | VSV # SV | # RSV |
|-------|----------|---------|--------|-----------|----------|-------|
| 0 | 17 | 15 | 18 | 1206 | 2938 | 59 |
| 1 | 15 | 13 | 12 | 757 | 1887 | 38 |
| 2 | 34 | 23 | 30 | 2183 | 5015 | 100 |
| 3 | 32 | 21 | 27 | 2506 | 4764 | 95 |
| 4 | 30 | 30 | 35 | 1784 | 3983 | 80 |
| 5 | 29 | 23 | 27 | 2255 | 5235 | 105 |
| 6 | 30 | 18 | 24 | 1347 | 3328 | 67 |
| 7 | 43 | 39 | 57 | 1712 | 3968 | 79 |
| 8 | 47 | 35 | 40 | 3053 | 6978 | 140 |
| 9 | 56 | 40 | 40 | 2720 | 6348 | 127 |

purposes only (to see the maximum possible effect of varying the threshold), the Bayes error computed on the *test* set.

Table 3 compares results on the test set for the three systems, where the Bayes threshold (computed with the training set) was used for RS. The results for all ten digits combined are 1.4% error for ORIG, 1.0% for VSV (with roughly twice as many multiply adds) and 1.1% for RS (with a factor of 22 fewer multiply adds than ORIG).

The reduced set conjugate gradient algorithm does not reduce the objective function $\rho^2$ (Equation (7)) to zero. For example, for the first 5 digits, $\rho^2$ is only reduced on average by a factor of 2.4 (the algorithm is stopped when progress becomes too slow). It is striking that nevertheless, good results are achieved.

# 6   DISCUSSION

The only systems in LeCun et al (1995) with better than 1.1% error are LeNet5 (0.9% error, with approximately 350K multiply-adds) and boosted LeNet4 (0.7% error, approximately 450K multiply-adds). Clearly SVMs are not in this league yet (the RS system described here requires approximately 650K multiply-adds).

However, SVMs present clear opportunities for further improvement. (In fact, we have since trained a VSV system with 0.8% error, by choosing a different kernel). More invariances (for example, for the pattern recognition case, small rotations, or varying ink thickness) could be added to the virtual support vector approach. Further, one might use only those virtual support vectors which provide new information about the decision boundary, or use a measure of such information to keep only the most important vectors. Known invariances could also be built directly into the SVM objective function.

Viewed as an approach to function approximation, the reduced set method is currently restricted in that it assumes a decision function with the same functional form as the original SVM. In the case of quadratic kernels, the reduced set can be computed both analytically and efficiently (Burges, 1996). However, the conjugate gradient descent computation for the general kernel is very inefficient. Perhaps re-

laxing the above restriction could lead to analytical methods which would apply to more complex kernels also.

## Acknowledgements

We wish to thank V. Vapnik, A. Smola and H. Drucker for discussions. C. Burges was supported by ARPA contract N00014-94-C-0186. B. Schölkopf was supported by the Studienstiftung des deutschen Volkes.

## Footnotes

*Part of this work was done while B.S. was with AT&T Research, Holmdel, NJ.

## References

[1] Boser, B. E., Guyon, I. M., Vapnik, V., *A Training Algorithm for Optimal Margin Classifiers*, Fifth Annual Workshop on Computational Learning Theory, Pittsburgh ACM (1992) 144–152.

[2] Bottou, L., Cortes, C., Denker, J. S., Drucker, H., Guyon, I., Jackel, L. D., Le Cun, Y., Müller, U. A., Säckinger, E., Simard, P., Vapnik, V., *Comparison of Classifier Methods: a Case Study in Handwritten Digit Recognition*, Proceedings of the 12th International Conference on Pattern Recognition and Neural Networks, Jerusalem (1994)

[3] Burges, C. J. C., *Simplified Support Vector Decision Rules*, 13th International Conference on Machine Learning (1996), pp. 71 – 77.

[4] Cortes, C., Vapnik, V., *Support Vector Networks*, Machine Learning **20** (1995) pp. 273 – 297

[5] LeCun, Y., Jackel, L., Bottou, L., Brunot, A., Cortes, C., Denker, J., Drucker, H., Guyon, I., Müller, U., Säckinger, E., Simard, P., and Vapnik, V., *Comparison of Learning Algorithms for Handwritten Digit Recognition*, International Conference on Artificial Neural Networks, Ed. F. Fogelman, P. Gallinari, pp. 53-60, 1995.

[6] Schölkopf, B., Burges, C.J.C., Vapnik, V., *Extracting Support Data for a Given Task*, in Fayyad, U. M., Uthurusamy, R. (eds.), Proceedings, First International Conference on Knowledge Discovery & Data Mining, AAAI Press, Menlo Park, CA (1995)

[7] Schölkopf, B., Burges, C.J.C., Vapnik, V., *Incorporating Invariances in Support Vector Learning Machines*, in Proceedings ICANN'96 — International Conference on Artificial Neural Networks. Springer Verlag, Berlin, (1996)

[8] Simard, P., Victorri, B., Le Cun, Y., Denker, J., *Tangent Prop — a Formalism for Specifying Selected Invariances in an Adaptive Network*, in Moody, J. E., Hanson, S. J., Lippmann, R. P., *Advances in Neural Information Processing Systems 4*, Morgan Kaufmann, San Mateo, CA (1992)

[9] Vapnik, V., *Estimation of Dependences Based on Empirical Data*, [in Russian] Nauka, Moscow (1979); English translation: Springer Verlag, New York (1982)

[10] Vapnik, V., *The Nature of Statistical Learning Theory*, Springer Verlag, New York (1995)

[11] Vapnik, V., Golowich, S., and Smola, A., *Support Vector Method for Function Approximation, Regression Estimation, and Signal Processing*, Submitted to Advances in Neural Information Processing Systems, 1996

[12] Vetter, T., Poggio, T., and Bülthoff, H., *The Importance of Symmetry and Virtual Views in Three–Dimensional Object Recognition*, Current Biology **4** (1994) 18–23